# Strong Unimodality and Exact Learning of Constant Depth $\mu$-Perceptron Networks

**Mario Marchand**
Department of Computer Science
University of Ottawa
Ottawa, Ont., Canada K1N 6N5
marchand@csi.uottawa.ca

**Saeed Hadjifaradji**
Department of Physics
University of Ottawa
Ottawa, Ont., Canada K1N 6N5
saeed@physics.uottawa.ca

## Abstract

We present a statistical method that exactly learns the class of constant depth $\mu$-perceptron networks with weights taken from $\{-1, 0 + 1\}$ and arbitrary thresholds when the distribution that generates the input examples is member of the family of product distributions. These networks (also known as nonoverlapping perceptron networks or read-once formulas over a weighted threshold basis) are loop-free neural nets in which each node has only one outgoing weight. With arbitrary high probability, the learner is able to exactly identify the connectivity (or skeleton) of the target $\mu$-perceptron network by using a new statistical test which exploits the strong unimodality property of sums of independent random variables.

## 1 INTRODUCTION

From a computational learning theory perspective, it is well known that efficient learning of non trivial (or non simple) neural network function classes is possible only when either (1) the learner is able to use membership queries or (2) the distribution that generates the input examples is not arbitrary but member of some well defined family. Following several positive learnability results on different classes of *read-once* Boolean formulas, a membership query algorithm has been recently proposed [4] for learning the class of *nonoverlapping* perceptron networks. These networks (also known as $\mu$-perceptron networks or read-once formulas over a weighted threshold basis) are loop-free neural nets in which each node has only one outgoing weight. If membership queries are not permitted (as we assume throughout this paper), learning this class becomes intractable [6] under arbitrary input distribu-

tions. However, under the *uniform* distribution, a PAC learning algorithm has been proposed recently [2] for a quite restricted subclass called *generalized μ-perceptron decision lists*. As an important step towards the learnability of the whole class of μ-perceptron networks under "simple" distributions, we present in this paper a statistical method that exactly learns the class of constant depth μ-perceptron networks under the family of *product distributions*, *i.e.* distributions in which the setting of each input variable is chosen independently of the other variables. Eventhough the depth of the network must be fixed to a constant, we satisfy here a harder learning criterion than the one proposed by the PAC model [9]. Indeed, with arbitrary high probability, the proposed algorithm is able to *exactly identify* [1] the target function. Moreover, because of its statistical nature [7], the proposed algorithm can tolerate a classification noise rate $\eta$ up to the information theoretic limit of $\eta = 1/2$.

There exist other statistical methods to learn other classes of read-once formulas under particular distributions [1] and product distributions [8]. They all basically differ in the statistical tests they use to identify the gate parameters and the formula's skeleton. Our key novel contribution is to introduce a new test (for discovering the network's connectivity) which exploits the *strong unimodality* [5] property of sums of independent random variables.

## 2   DEFINITIONS

We consider the problem of learning Boolean functions of the Boolean domain $\{0, 1\}^n$. Let $X = \{x_1, x_2 \cdots, x_n\}$ be the set of $n$ input variables and $\mathbf{x} \in \{0, 1\}^n$ be some assignment of these $n$ variables, we denote by $\mathbf{x}_V$ the restriction of assignment $\mathbf{x}$ on the variables in $V \subseteq X$. A *perceptron* $g$ on $V$ is defined by a vector of $v = |V|$ weights $w_i$ and a single threshold $\theta$. As usual, for any $\mathbf{x}_V \in \{0, 1\}^v$, the output of $g(\mathbf{x}_V)$ is 1 whenever $\sum_{i \in V} w_i x_i > \theta$ and 0 otherwise.

We restrict ourselves to the case where each $w_i \in \{-1, 0, +1\}$ but the thresholds are arbitrary so that, without loss of generality (w.l.o.g.), $\theta \in \{-v - 1, \cdots v\}$. A perceptron is said to be *positive* if all its incoming weights are $+1$. The learning algorithm will use the following classification for positive perceptrons.

*T1 perceptrons:* These are perceptrons which output 1 iff one or more of its inputs are set to 1. These are OR gates of multiple inputs.

*T0 perceptrons:* These are perceptrons which output 0 iff one or more of its inputs are set to 0. These are AND gates of multiple inputs.

*T11 perceptrons:* These are perceptrons which output 1 iff two or more of its inputs are set to 1. These include majority gates of three inputs.

*T00 perceptrons:* These are perceptrons which output 0 iff two or more of its inputs are set to 0. These include majority gates of four variables.

*TG perceptrons:* All the perceptrons which do not belong to any one of the above four categories. They must, therefore, have at least five inputs.

Each perceptron can have variables and/or other perceptrons as inputs. Hence, a *node* will denote either a variable or a perceptron. The class of *μ-perceptron networks* is the set of all Boolean functions that can be represented as a loop-free network of perceptrons where each node (including input units) has only one outgoing weight. The output unit of a network will often be referred as the *root node*. We say that a node is a *child* of the *parent* perceptron $g$ if it is an immediate input to perceptron $g$. Children of the same perceptron are called *siblings*. A perceptron is said to be a *bottom level perceptron* if all its children are variables. The *depth* of a node is defined as the number of perceptrons (including the parent of the node and the root node) on the path from the parent of the node to the root. The perceptrons

on this path are called the *ancestors* of that node. The *descendants* of perceptron $g$, denoted by $\mathrm{desc}(g)$, is the set of nodes that have perceptron $g$ as an ancestor. The depth of a network is defined as the depth of the deepest variable in the net. The *least common ancestor* of a set $V$ of nodes, denoted by $\mathrm{lca}(V)$, is defined as the deepest ancestor which is common to every node in $V$. Variables $\{x_i, x_j, x_k\}$ are said to *meet* at perceptron $g$, *iff* $\mathrm{lca}(x_i, x_j) = \mathrm{lca}(x_i, x_k) = \mathrm{lca}(x_j, x_k) = g$. If there does not exist a perceptron $g$ having this property, then variables $\{x_i, x_j, x_k\}$ are said not to meet.

In this paper, we use a learning criterion which is more ambitious than the PAC criterion introduced by Valiant [9]. We consider that each training example **x** is generated by an unknown product distribution $D$ on $\{0,1\}^n$ and then labeled according to an unknown target Boolean function $f$ representable as a $\mu$-perceptron network. After observing a set of such examples, the goal of the learning algorithm is to produce an hypothesis function $h$ which is the exact equivalent of $f$. More formally we say that algorithm $A$ *exactly learns* (or exactly identifies) a class $F$ of of Boolean functions *iff* for any $0 < \delta < 1$, any product distribution $D$ on $\{0,1\}^n$, and any target function $f \in F$, algorithm $A$ outputs, with probability at least $1 - \delta$ an hypothesis function $h$ such that $h(\mathbf{x}) = f(\mathbf{x}) \ \forall \ \mathbf{x} \in \{0,1\}^n$.

The learning algorithm will perform several statistical tests to build its hypothesis. Namely, for each variable $x_i$, it will estimate its *influence*, defined as:

$$\mathrm{Infl}(x_i) \overset{\text{def}}{=} \Pr(f = 1 | x_i = 1) - \Pr(f = 1 | x_i = 0) \tag{1}$$

where all probabilities (here and in the sequel) are defined with respect to the (unknown) training product distribution $D$. The empirical estimate of $\Pr(A)$ will be denoted as $\hat{\Pr}(A)$. We will also use, $\mathrm{Infl}_g(x_i)$ to denote the influence of $x_i$ on the subformula of $f$ which is rooted at perceptron $g$. Also, $\mathrm{Infl}(x_i | x_j = a)$ will denote the influence of $x_i$ given that variable $x_j$ is fixed to value $a$. To discover the skeleton of the target function, the learner will compute the *coinfluence* of several triples of variables, defined as:

$$C^k_{i,j} \overset{\text{def}}{=} \frac{\mathrm{Infl}(x_j | x_i = 1, x_k = 0)}{\mathrm{Infl}(x_j | x_i = 0, x_k = 0)} - \frac{\mathrm{Infl}(x_j | x_i = 1, x_k = 1)}{\mathrm{Infl}(x_j | x_i = 0, x_k = 1)} \tag{2}$$

Because $h$ must make zero error with $f$, the learner must produce an hypothesis $h$ which contains all the input variables and all the perceptrons of $f$ (except those variables and perceptrons which are fixed to a constant value). Consequently, for a target $f$ defined on $n$ input variables $x_i$ and containing $r$ perceptrons $g_k$, we define $\epsilon_s$ as:

$$\epsilon_s \overset{\text{def}}{=} \min \left\{ \Pr(x_i = a)_{i \in \{0, \cdots n\}, a \in \{0,1\}}, \Pr(g_k = b)_{k \in \{0, \cdots r\}, b \in \{0,1\}} \right\} \tag{3}$$

Hence, $\forall \ i \in \{0, \cdots n\}$ we have: $\epsilon_s \leq \Pr(x_i = 1) \leq 1 - \epsilon_s$ and $\forall \ k \in \{0, \cdots r\}$ we have: $\epsilon_s \leq \Pr(g_k = 1) \leq 1 - \epsilon_s$. To exactly learn the class of constant depth $\mu$-percepton networks, the proposed algorithm needs a number of examples which is polynomial in $1/\epsilon_s$ (see the algorithm **LearnNPN**).

## 3    THE LEARNING ALGORITHM

We first perform some simplifying reductions that hold for any target $\mu$-perceptron net $f$. (1) We can assume, w.l.o.g., that only input variables have a negative outgoing weight. Indeed, if a perceptron $g$ has a $-1$ outgoing weight, we can replace it by a perceptron which has all its incoming weights negated and a $+1$

outgoing weight; this leaves the computation by $f$ unchanged when we add $+1$ to the threshold of $g$'s parent. In this manner, all $-1$ weights are pushed to the input variables. (2) $T1$ perceptrons do not have $T1$ perceptrons as children since such nodes can always be merged. The same remark is true for $T0$ perceptrons. (3) Because the output of each node is Boolean valued, each perceptron has at least two inputs. This implies that $f$ has at most $n - 1$ perceptrons.

The first step of the algorithm is to identify the weight $w_i$ that springs out of each input variable $x_i$. For this we appeal to the following lemma:

**Lemma 1** *Let $f$ be any μ-perceptron network with weights taken from $\{-1, 0, +1\}$ and arbitrary thresholds. Let $D$ be any product distribution on $\{0, 1\}^n$. Let $g$ be any perceptron with $v$ weights and for which $\rho \leq \Pr(g = 1) \leq 1 - \rho$. Let input variable $x_i$ be a child of $g$. Then:*

$$\mathrm{Infl}_g(x_i) \begin{cases} > & +\rho/(2v) & \text{if } w_i = +1 \\ = & 0 & \text{if } w_i = 0 \\ < & -\rho/(2v) & \text{if } w_i = -1 \end{cases}$$

*Moreover, if $x_i$ has depth $d$, then, we have:*

$$|\mathrm{Infl}(x_i)| > \left(\frac{\epsilon_s}{2n}\right)^d$$

Thus $\mathrm{Infl}(x_i)$ has a gap of $O([\epsilon_s/n]^d)$ that separates the three possible values for $w_i$. From Chernoff bounds [3], this implies that a sample size polynomial in $\epsilon_s/n$ is sufficient to find, with high probability, the *exact* value of $w_i$ when $d$ is fixed. After having identified all the weights in this manner, we transform the target function into its *positive form* simply by changing $x_i$ to $1 - x_i$ (and adding $+1$ to the threshold of $x_i$'s parent) whenever $w_i = -1$.

To find the skeleton of the target function, the algorithm will first find all the bottom level perceptrons (*i.e.* perceptrons whose children are all variables). Then, after finding the *exact* thresholds (for $TG$ perceptrons), we will consider these bottom level perceptrons as new "meta" variables (that replace their children) from which we can find their parent perceptrons. In this manner similar to Schapire's algorithm [8], we will build every perceptron of the net until we reach the root.

The coinfluence function will enable the learner to determine if certain variables are siblings of a perceptron $g$ and if $g$ is fed by other perceptrons. This is possible because the distribution of a sum of independent random variables is strongly unimodal [5]. More specifically, we have (and need) here a stronger property:

**Lemma 2** *Let $\{x_1, x_2 \cdots, x_v\}$ be $v$ independent random Boolean variables, each with $\Pr(x_i = 1) = q_i$ and let $S \overset{\text{def}}{=} \sum_{i=1}^{v} x_i$. Then for any $\{q_1 \cdots, q_v\}$ and any $k \in \{1, \cdots, v\}$:*

$$\frac{\Pr(S = k - 1)}{\Pr(S = k)} - \frac{\Pr(S = k - 2)}{\Pr(S = k - 1)} \geq \frac{\Pr(S = 0)}{\Pr(S = 1)} = \frac{1}{v < \alpha >_v}$$

*where $\alpha_i \overset{\text{def}}{=} q_i/(1 - q_i)$ and $< \alpha >_v \overset{\text{def}}{=} \sum_{i=1}^{v} \alpha_i/v$.*

**Proof:** Omitted from this abstract but one can easily verify its exactness in the case where $q_i = q$ for all $i = 1 \cdots, v$. □.

The next lemma constitutes our main tool for finding the connectivity of $f$. It is expressed in terms of what we call the *strong unimodal gap* $\gamma_n$:

$$\gamma_n \overset{\text{def}}{=} \min\left\{\frac{1}{n < \alpha >_n}, \frac{1}{n < 1/\alpha >_n}\right\}$$

where: $< 1/\alpha >_n \stackrel{\text{def}}{=} \sum_{i=1}^{n} \alpha_i^{-1}/n$.

**Lemma 3** *Let $\{x_i, x_j, x_k\}$ be any triple of variables such that each is a child of some TG perceptron. Then:*

*1.* $C_{i,j}^k \geq \gamma_n$ *if $\{x_i, x_j, x_k\}$ are siblings of a perceptron $g$.*

*2.* $C_{i,j}^k = 0$ *if $x_k \notin \text{desc}(\text{lca}(x_i, x_j))$*

*3.* $C_{i,j}^k = C_{i,l}^k$ *if $\{x_i, x_j, x_k\}$ meet at a perceptron $g$ that has a perceptron $g_j$ as a child with the property that both $x_j$ and $x_l$ feed $g_j$.*

*4.* **If** $\{ C_{i,j}^k \geq \gamma_n$ *and $x_l$ feeds $\text{lca}(x_i, x_j)$ through a perceptron $g_l$ which is not fed by $(x_i, x_j) \}$* **Then** $C_{i,j}^l > \gamma_n^3 \cdot \text{Infl}(x_l)$

**Proof sketch:** If $\{x_i, x_j, x_k\}$ are siblings of a perceptron $g$ of threshold $\theta$, then $C_{i,j}^k = [\Pr(S = \theta - 1)/\Pr(S = \theta)] - [\Pr(S = \theta - 2)/\Pr(S = \theta - 1)]$ which, from lemma 2, establishes fact 1. Let $g = \text{lca}(x_i, x_j)$. Then, if $x_k \notin \text{desc}(\text{lca}(x_i, x_j))$, $\text{Infl}(x_j | x_i = a, x_k = b) = \text{Infl}(g | x_k = b) \cdot \text{Infl}_g(x_j | x_i = a)$ which establishes fact 2. The proofs for fact 3 and 4 are omitted from this abstract. $\square$

The constraints on $x_i$ and $x_k$ in lemma 3 are to avoid vanishing denominators in $C_{i,j}^k$. This does not create any problems since by using simpler tools, we can always find the children of the $T0, T1, T00, T11$ perceptrons before those of the $TG$ perceptrons. In the following we also explain how to identify the non-$TG$ bottom level perceptrons.

**Lemma 4** *Variable $x_i$ is a child of a T1 perceptron iff there exist $x_j$ such that $\text{Infl}(x_j | x_i = 1) = 0$. Otherwise $\text{Infl}(x_j | x_i = 1) > \text{Infl}(x_j) \cdot \gamma_n$ for all $x_j \neq x_i$.*

*Moreover, a set $W$ of variables, each of which is a child of a T1 perceptron, is a set of siblings iff $\text{Infl}(x_j | x_i = 1) = 0 \; \forall \; \{x_i, x_j\} \in W$.*

*Moreover, If $\{ W \subseteq V$ is a set of variables, all siblings of a T1 perceptron $g$, such that no children of $g$ is in $V - W \}$ Then $\{ g$ is a bottom level perceptron with respect to $V$ iff $\text{Infl}(x_k | x_i = 1) > \text{Infl}(x_k) \cdot \gamma_n$ for all $x_k \in V - W, x_i \in W$. Otherwise there exist $x_k \in V - W$ and $x_i \in W$ such that $\text{Infl}(x_k | x_i = 1) = 0 \}$.*

*The lemma is valid when we replace T1 by T0 if the condition $x_i = 1$ is replaced by $x_i = 0$.*

**Proof idea:** Directly follows from lemma 2, the definitions of $T1$ and $T0$ and from the fact that no two consecutive $T1$ (nor $T0$) perceptrons occur in $f$. $\square$

From this lemma, we define a routine, **Find-bl-T1**$(V)$, that finds all $T1$ perceptrons which are bottom level with respect to the set $V$ of variables (or meta variables). It achieves this by testing, for each pair of variables, if $\hat{\text{Infl}}(x_j | x_i = 1)/\hat{\text{Infl}}(x_j) > \gamma_n/2$. (By using Chernoff bounds, we find the probability of making the correct decision for each variable as a function of the sample size $m$.) Moreover, the output of this routine is a set $V'$ which consists of the original set $V$ from which the siblings have been replaced by their bottom level $T1$ parents (with their children connected) as new meta variables. It also tags those variables in $V$ that are children of some non-bottom level $T1$ perceptron. This is to warn the subsequent routines of not using these variables to find out if they are children of other types of perceptrons. An identical definition and a similar operation applies for **Find-bl-T0**$(V)$. The same applies also for **Find-bl-T11**$(V)$, **Find-bl-T00**$(V)$ and **Find-bl-TG**$(V)$ but for them, we need to use the following lemmas.

**Lemma 5** *Let $x_i$ and $x_j$ be variables which neither is a child of a $T1$ or a $T0$ perceptron. Then $\{x_i, x_j\}$ are siblings of a $T11$ perceptron iff there exist $x_k$ such that $\mathrm{Infl}(x_k|x_i = x_j = 1) = 0$. Otherwise $\mathrm{Infl}(x_k|x_i = x_j = 1) > \mathrm{Infl}(x_k) \cdot \gamma_n^2$ for all $x_k \notin \{x_i, x_j\}$.*

*Moreover, let $V$ be a set of variables for which no one is a child of some $T0$ or $T1$ perceptron. Let $W \subseteq V$ be a set of variables, all siblings of a $T11$ perceptron $g$ and such that no children of $g$ is in $V - W$. Then $g$ is a bottom level perceptron with respect to $V$ iff $C_{i,j}^k = 0$ for all $x_k \in V - W$ and $\{x_i, x_j\} \in W$. Otherwise there exist $x_k \in V - W$ and $\{x_i, x_j\} \in W$ such that $C_{i,j}^k > \mathrm{Infl}(x_k) \cdot \gamma_n^3$.*

*The lemma is valid when we replace $T11$ by $T00$ if the condition $x_i = x_j = 1$ is replaced by $x_i = x_j = 0$.*

**Proof idea:** Follows from lemmas 2 and 3 and from the definitions of $T11$ and $T00$ perceptrons. □

**Lemma 6** *Let $V$ be a set of variables, each of which is a child of some $TG$ perceptron. Let $W \subseteq V$ be a set of variables for which $C_{i,j}^k \geq \gamma_n \; \forall \; \{x_i, x_j, x_k\} \in W$. Then $W$ is a set of siblings of a bottom level $TG$ perceptron $g$ (and thus $g$ is bottom level with respect to $V$) iff there does not exist any $\{l, m\} \in V - W$ and $\{i, j\} \in W$ for which all of these properties hold:*

1. *$C_{i,j}^l > \gamma_n^3 \cdot \mathrm{Infl}(x_l) \quad and \quad C_{i,j}^m > \gamma_n^3 \cdot \mathrm{Infl}(x_m)$*

2. *$C_{l,m}^i = C_{l,m}^j = 0$*

*Moreover, the threshold $\theta$ of a bottom level $TG$ perceptron $g$ (in positive form) is obtained by the value of $k$ for which $\Pr(f = 1|S = k+1) - \Pr(f = 1|S = k) \geq \mathrm{Infl}(x_j)$ where $x_j$ can be any child of $g$ and $S$ denotes the sum over all its children. This difference is zero if $k \neq \theta$.*

Having sketched the action of the different **Find-bl-T\*** routines, we now propose the following learning algorithm.

<div align="center">

**Algorithm LearnNPN**$(n, \epsilon_s, \delta)$

</div>

1. Call $m = \left( \dfrac{64}{\epsilon_s^2 \gamma_n^5} \right)^2 \left( \dfrac{2n}{\epsilon_s} \right)^{4d} \ln \left( \dfrac{32dn^3}{\delta} \right)$ training examples.

2. For every $x_i \in X$, let $w_i = +1$ if $\widehat{\mathrm{Infl}}(x_i) > \epsilon_s/4n$, let $w_i = -1$ if $\widehat{\mathrm{Infl}}(x_i) < -\epsilon_s/4n$ and let $w_i = 0$ otherwise. Let $x_i = 1 - x_i$ whenever $w_i = -1$ (conversion into the positive form). Let $V = X$.

3. Repeat {
         Repeat {
               Repeat {
                     Repeat {Let $V_i = V$; $V = $ **Find-bl-T1(Find-bl-T0**$(V_i)$**)**
                     } Until $V = V_i$
                     Let $V_i = V$; $V = $ **Find-bl-T00**$(V_i)$
               } Until $V = V_i$
               Let $V_i = V$; $V = $ **Find-bl-T11**$(V_i)$
         } Until $V = V_i$
         $V = $ **Find-bl-TG**$(V)$
   } Until only one meta-variable remains in $V$

4. Return this meta-variable (with all the others attached to it) as our hypothesis network $h$ and convert from positive to normal form.

The nested loops insure that, every time the set $V$ of meta-variables is updated, all bottom level $T0$ and $T1$ perceptrons are found before the $T00$ and $T11$ perceptrons which are themselves found before the $TG$ perceptrons. This is essential in order that the **Find-bl-T\*** routines make proper use of lemma 5 and 6.

**Theorem 1** *Under product distributions, the algorithm* **LearnNPN** *exactly learns the class of $\mu$-perceptrons networks of depth at most $d$ with weights taken from $\{-1, 0, +1\}$ and arbitrary thresholds. The algorithm runs in time of $O(m \times d \times n^3)$.*

**Proof idea:** By using Chernoff bounds [3], one can verify that the above sample of $m$ examples is sufficient to ensure that all probabilities are estimated with enough precision to have $h(\mathbf{x}) = f(\mathbf{x})$ $\forall \mathbf{x} \in \{0, 1\}^n$ with probability at least $1 - \delta$.

## Acknowledgments

We thank Mostefa Golea and Hans U. Simon for useful comments and discussions about technical points. M. Marchand is supported by NSERC grant OGP0122405. Saeed Hadjifaradji is supported by the MCHE of Iran.

## References

[1] Goldman S., Kearns M.J., & Schapire R., (1990). "Exact identification of circuits using fixed points of amplification functions. *Proceedings of the 31st Symposium on Foundations of Computer Science*, Los Alamitos, CA: IEEE Computer Society Press.

[2] Golea M., Marchand M., & Hancock T.R., (1995) "On Learning $\mu$-Perceptron Networks On the Uniform Distribution", to appear in *Neural Networks*. For a short version see: *Advances in Neural Information Processing Systems 5*, pp. 591–598, San Mateo CA, Morgan Kaufmann Publishers, (1993). Also: *Computational Learning Theory: EuroCOLT'93*, pp. 47–60, Oxford University Press, (1994).

[3] Hagerup T. & Rub C. (1989) "A Guided Tour to Chernoff Bounds", *Info. Proc. Lett.*, Vol. 33, 305–308.

[4] Hancock T.R., Golea M., & Marchand M., (1994) "Learning Nonoverlapping Perceptron Networks from Examples and Membership Queries", *Machine Learning*, vol. 16, pp. 161–183.

[5] Ibragimov I.A., (1956) "On the composition of unimodal distributions", *Theor. Probability Appl.* vol. 1, 255–266. *Also:* Keilson J. & Gerber H., (1971) "Some results for discrete unimodality", *J. Amer. Statist. Assoc.*, vol. 66, 386–389.

[6] Kearns M.J., Li M., Pitt L. & Valiant L. G., (1987). "On the learnability of boolean formulae" *Proceedings of the Nineteenth Annual ACM Symposium on Theory of Computing* (pp. 285–295). New York: ACM Press.

[7] Kearns M. (1993). "Efficient Noise-Tolerant Learning from Statistical Queries", *Proceedings of the Twenty Fifth Annual ACM Symposium on the Theory of Computation*, p. 392.

[8] Schapire R., (1994) "Learning probabilistic read-once formulas on product distributions" *Machine Learning* vol. 14, 47–81. San Mateo, CA: Morgan Kaufman.

[9] Valiant L.G. (1984) "A Theory of the Learnable", *Comm. ACM*, Vol. 27, 1134–1142.
